# ICA based on a Smooth Estimation of the Differential Entropy

**Lev Faivishevsky**
School of Engineering, Bar-Ilan University
levtemp@gmail.com

Jacob Goldberger
School of Engineering, Bar-Ilan University
goldbej@eng.biu.ac.il

## Abstract

In this paper we introduce the MeanNN approach for estimation of main information theoretic measures such as differential entropy, mutual information and divergence. As opposed to other nonparametric approaches the MeanNN results in smooth differentiable functions of the data samples with clear geometrical interpretation. Then we apply the proposed estimators to the ICA problem and obtain a smooth expression for the mutual information that can be analytically optimized by gradient descent methods. The improved performance of the proposed ICA algorithm is demonstrated on several test examples in comparison with state-of-the-art techniques.

## 1   Introduction

Independent component analysis (ICA) is the problem of recovering latent random vector from observations of unknown linear functions of that vector. Assume a data $S \in R^d$ is generated via $d$ independent sources. We observe $X = AS$ where $A$ is an unknown square matrix called the mixing matrix. We are given repeated observation dataset $\{x_1, ..., x_n\}$ and our goal is to recover the linear transformation $A$ and the sources $s_1, ..., s_n$ that generated our data $x_i = As_i$.

Given the minimal statement of the problem, it has been shown [6] that one can recover the original sources up to a scaling and a permutation provided that at most one of the underlying sources is Gaussian and the rest are non-Gaussian. Upon pre-whitening the observed data, the problem reduces to a search over rotation matrices in order to recover the source and mixing matrix in the sense described above [10]. We will assume henceforth that such pre-processing has been done. Specifying distributions for the components of $X$, one obtains a parametric model that can be estimated via maximum likelihood [3, 4]. Working with $W = A^{-1}$ as the parametrization, one readily obtains a gradient or fixed-point algorithm that yields an estimate $\hat{W}$ and provides estimates of the latent components via $\hat{S} = \hat{W}X$ [10].

In practical applications the distributions of the $d$ components of $X$ are unknown. Therefore it is preferable to consider the ICA model as a semiparametric model in which the distributions of the components of $X$ are left unspecified. The problem is then, obviously, to find a suitable *contrast function*, i.e. a target function to be minimized in order to estimate the ICA model. The earliest ICA algorithms were based on contrast functions defined in terms of expectations of a single fixed nonlinear function, chosen in ad-hoc manner [5]. More sophisticated algorithms have been obtained by careful choice of a single fixed nonlinear function, such that the expectations of this function yield a robust approximation to the mutual information [9].

Maximizing the likelihood in the semiparametric ICA model is essentially equivalent to minimizing the mutual information between the components of the estimate $\hat{S} = \hat{W}X$ [4]. The usage of the mutual information as a contrast function to be minimized in estimating the ICA model is well motivated, quite apart from the link to maximum likelihood [6].

Estimating MI from a given finite sample set is difficult. Several modern approaches rely on $k$-nearest neighbor estimates of entropy and mutual information [12, 16]. Recently the Vasicek estimator [17] for the differential entropy of 1D random variables, based on $k$-nearest neighbors statistics, was applied to ICA [8, 13]. In addition ICA was studied by another recently introduced MI estimator [16]. However, the derivative of the estimators that are based on order statistics can hardly be computed and therefore the optimization of such numerical criteria can not be based on gradient techniques. Also the result numerical criteria tend to have a non-smooth dependency on sample values. The optimization therefore should involve computation of contrast function on a whole grid of searched parameters.

In addition, such estimators do not utilize optimally the whole amount of data included in the samples of random vectors. Therefore they require significant artificial enlargement of data sets by a technique called data augmentation [13] that replaces each data point in sample with R-tuple (R is usually 30) of points given by an statistical procedure with ad-hoc parameters. An alternative is the Fourier filtering of the estimated values of the evaluated MI estimators [16].

In the present paper we propose new smooth estimators for the differential entropy, the mutual information and the divergence. The estimators are obtained by a novel approach averaging $k$-nearest neighbor statistics for the all possible values of order statistics $k$. The estimators are smooth, their derivatives may be easily analytically calculated thus enabling fast gradient optimization techniques. They fully utilize the amount of data comprised into a random variable sample. The estimators provide a novel geometrical interpretation for the entropy. When applied to ICA problem, the proposed estimator leads to the most precise results for many distributions known at present.

The rest of the paper is organized as follows: Section 2 reviews the $k$NN approach for the entropy and divergence estimation, Section 3 introduces the mean estimator for the differential entropy, the mutual information and the divergence. Section 4 describes the application of the proposed estimators to the ICA problem and Section 5 describes conducted numerical experiments.

## 2 $k$NN Estimators for the Differential Entropy

We review the nearest neighbor technique for the Shannon entropy estimation. The differential entropy of $X$ is defined as:

$$H(X) = -\int f(x) \log f(x) dx \tag{1}$$

We describe the derivation of the Shannon differential entropy estimate of [11, 18]. Our aim is to estimate $H(X)$ from a random sample $(x_1, ..., x_n)$ of $n$ random realizations of a $d$-dimensional random variable $X$ with unknown density function $f(x)$. The entropy is the average of $-\log f(x)$. If one had unbiased estimators for $\log f(x_i)$, one would arrive to an unbiased estimator for the entropy. We will estimate $\log f(x_i)$ by considering the probability density function $P_{ik}(\epsilon)$ for the distance between $x_i$ and its $k$-th nearest neighbor (the probability is computed over the positions of all other $n-1$ points, with $x_i$ kept fixed). The probability $P_{ik}(\epsilon)d\epsilon$ is equal to the chance that there is one point within distance $r \in [\epsilon, \epsilon + d\epsilon]$ from $x_i$, that there are $k-1$ other points at smaller distances, and that the remaining $n-k-1$ points have larger distances from $x_i$. Denote the mass of the $\epsilon$-ball centered at $x_i$ by $p_i(\epsilon)$, i.e. $p_i(\epsilon) = \int_{\|x-x_i\|<\epsilon} f(x)dx$. Applying the trinomial formula we obtain:

$$P_{ik}(\epsilon) = \frac{(n-1)!}{1!(k-1)!(n-k-1)!} \frac{dp_i(\epsilon)}{d\epsilon} p_i^{k-1}(1-p_i)^{n-k-1} \tag{2}$$

It can be easily verified that indeed $\int P_{ik}(\epsilon)d\epsilon = 1$. Hence, the expected value of the function $\log p_i(\epsilon)$ according to the distribution $P_{ik}(\epsilon)$ is:

$$E_{P_{ik}(\epsilon)}(\log p_i(\epsilon)) = \int_0^\infty P_{ik}(\epsilon) \log p_i(\epsilon) d\epsilon = k \binom{n-1}{k} \int_0^1 p^{k-1}(1-p)^{n-k-1} \log p \, dp \tag{3}$$

$$= \psi(k) - \psi(n)$$

where $\psi(x)$ is the digamma function (the logarithmic derivative of the gamma function). To verify the last equality, differentiate the identity $\int_0^1 x^{a-1}(1-x)^{b-1} = \Gamma(a)\Gamma(b)/\Gamma(a+b)$ with respect to

the parameter $a$ and recall that $\Gamma'(x) = \psi(x)\Gamma(x)$. The expectation is taken over the positions of all other $n - 1$ points, with $x_i$ kept fixed. Assuming that $f(x)$ is almost constant in the entire $\epsilon$-ball around $x_i$, we obtain:

$$p_i(\epsilon) \approx c_d \epsilon^d f(x_i). \tag{4}$$

where $d$ is the dimension of $x$ and $c_d$ is the volume of the $d$-dimensional unit ball ($c_d = \pi^{d/2}/\Gamma(1 + d/2)$ for Euclidean norm). Substituting Eq. (4) into Eq. (3), we obtain:

$$-\log f(x_i) \approx \psi(n) - \psi(k) + \log(c_d) + dE(\log(\epsilon)) \tag{5}$$

which finally leads to the unbiased $k$NN estimator for the differential entropy [11]:

$$H_k(X) = \psi(n) - \psi(k) + \log(c_d) + \frac{d}{n}\sum_{i=1}^{n}\log \epsilon_i \tag{6}$$

where $\epsilon_i$ is the distance from $x_i$ to its $k$-th nearest neighbor. An alternative proof of the asymptotic unbiasedness and consistency of the $k$NN estimator is found at [15].

A similar approach can be used to obtain a $k$NN estimator for the Kullback-Leibler divergence [19]. The estimator works as follows. Let $\{x_1, ..., x_n\}$ and $\{y_1, ..., y_m\}$ be i.i.d. $d$-dimensional samples drawn independently from the densities $p$ and $q$ respectively. By definition the divergence is given by:

$$D(p\|q) = \int p(x)\log\frac{p(x)}{q(x)} \tag{7}$$

The distance of $x_i$ to its nearest neighbor in $\{x_j\}_{j\neq i}$ is defined as

$$\rho_n(i) = \min_{j\neq i} d(x_i, x_j) \tag{8}$$

We also define the distance of $x_i$ to its nearest neighbor in $\{y_j\}$

$$\nu_n(i) = \min_{j=1,...,m} d(x_i, y_j) \tag{9}$$

Then the estimator of [19] is given by

$$\hat{D}_{n,m} = \frac{d}{n}\sum_{i=1}^{n}\log\frac{\nu_m(i)}{\rho_n(i)} + \log\frac{m}{n-1} \tag{10}$$

The authors established asymptotic unbiasedness and mean-square consistency of the estimator (10). The same proofs could be applied to obtain $k$-nearest neighbor version of the estimator:

$$\hat{D}_{n,m}^{k} = \frac{d}{n}\sum_{i=1}^{n}\log\frac{v_m^k(i)}{\rho_n^k(i)} + \log\frac{m}{n-1} \tag{11}$$

Being non-parametric, the $k$NN estimators (6, 11) rely on the order statistics. This makes the analytical calculation of the gradient hardly possible. Also it leads to a certain lack of smoothness of the estimator value as a function of the sample coordinates. One also should mention that finding the $k$-nearest neighbor is a computationally intensive problem. It becomes necessarily to use involved approximate nearest neighbor techniques for large data sets.

## 3 The MeanNN Entropy Estimator

We propose a novel approach for the entropy estimation as a function of sample coordinates. It is based on the fact that the $k$NN estimator (6) is valid for every $k$. Therefore the differential entropy can be also extracted from a mean of several estimators corresponding to different values of $k$. Next we consider all the possible values of order statistics $k$ from 1 to $n - 1$:

$$H_{mean} = \frac{1}{n-1}\sum_{k=1}^{n-1}H_k = \log(c_d) + \psi(n) + \frac{1}{n-1}\sum_{k=1}^{n-1}\left(-\psi(k) + \frac{d}{n}\sum_{i=1}^{n}\log \epsilon_{i,k}\right) \tag{12}$$

where $\epsilon_{i,k}$ is the $k$-th nearest neighbor of $x_i$. Consider the double-summation last term in Eq. (12). Exchanging the order of summation, the last sum adds for each sample point $x_i$ the sum of log of

its distances to all its nearest neighbors in the sample. It is of course equivalent to the sum of log of its distances to all other points in the sample set. Hence the mean estimator (12) for the differential entropy can be written as:

$$H_{mean} = \text{const} + \frac{d}{n(n-1)} \sum_{i \neq j} \log \|x_i - x_j\| \tag{13}$$

where the constant depends just on the sample size and dimensionality. We dub this estimator, the *MeanNN* estimator for differential entropy. It follows that the differential entropy (approximation) has a clear geometric meaning. It is proportional to log of the products of distances between each two points in a random i.i.d. sample. It is an intuitive observation since a higher entropy would lead to a larger scattering of the samples thus pairwise distances would grow resulting in a larger product of all distances. Moreover, the MeanNN estimator (13) is a smooth function of the sample coordinates. Its gradient can be easily found. The asymptotic unbiasedness and consistency of the estimator follow from the same properties of the $k$NN estimator (6). Obviously, the same method gives the mean estimator for the mutual information by usage of well known equality connecting the mutual information and marginal and joint entropies:

$$I_{mean}(X;Y) = H_{mean}(X) + H_{mean}(Y) - H_{mean}(X,Y) \tag{14}$$

We demonstrate the MeanNN estimator for the entropy in the case exponential distributed random variable $f(x, \mu) = \frac{1}{\mu} e^{-\frac{x}{\mu}}, x > 0, \mu > 0$. In this case case the entropy may be analytically calculated as $H = \log \mu + 1$. We compared the performance of the MeanNN estimator with $k$-nearest neighbor estimator (6) for various values of $k$. Results are given in Table 1. One may see that the mean square error of the MeanNN estimator is the same or worse for the traditional $k$NN estimators. But the standard deviation of the estimator values is best for the MeanNN estimator. Further we will apply MeanNN for optimization of a certain criterion based on the entropy. In such cases the most important characteristics of an estimator is its monotonic dependency on the estimated value and the prediction of the exact value of the entropy is less important. Therefore one may conclude that MeanNN is better applicable for optimization of entropy based numerical criteria.

|  | 1NN | 4NN | 10NN | MeanNN |
|---|---|---|---|---|
| Mean square error of entropy estimation | 0.0290 | 0.0136 | 0.0117 | 0.0248 |
| STD of estimator values | 0.1698 | 0.1166 | 0.1079 | 0.1029 |

Table 1: Performance of MeanNN entropy estimator in comparison with $k$NN entropy estimators. 100 samples of random variable, 10 various values of $\mu$ parameter, 100 repetitions.

To obtain the estimator for the divergence we apply the same mean approach to estimator (11) setting $m = n - 1$:

$$\hat{D}_{n,n-1}^{mean} = \frac{d}{n(n-1)} \sum_{k=1}^{n-1} \sum_{i=1}^{n} \log \frac{v_m^k(i)}{\rho_n^k(i)} = \frac{d}{n(n-1)} \left( \sum_{i,j} \log d(x_i, y_j) - \sum_{i \neq j} \log d(x_i, x_j) \right) \tag{15}$$

The mean estimator for the divergence has a clear geometric interpretation. If the product of all distances inside one sample is small in comparison with the product of pairwise distances between the samples then one concludes that divergence is large and vice versa.

## 4 The MeanNN ICA Algorithm

As many approaches do, we will use a contrast function

$$J(Y) = \int q(y_1, ..., y_d) \log \frac{q(y_1, .., y_d)}{\prod_{i=1}^{d} q(y_i)} d\mu = D(q(y_1, .., y_d) \| \prod_{i=1}^{d} q(y_i)) = \sum_{i=1}^{d} H(Y_i) - H(Y_1, ..., Y_d) \tag{16}$$

Considering $Y$ as linear function of $X$, $Y = WX$, it is easily verified [3, 7, 10] that

$$J(Y) = \sum_{t=1}^{d} H(Y_t) - H(X_1, ..., X_d) - \log(|W|) \tag{17}$$

In particular, the change in the entropy of the joint distribution under linear transformation is simply the logarithm of the Jacobian of the transformation. As we will assume the $X$'s to be pre-whitened, $W$ will be restricted to rotation matrices, therefore $\log(|W|) = 0$ and the minimization of $J(Y)$ reduces to finding

$$\hat{W} = \arg \min_{W} H(Y_1) + ... + H(Y_d) \tag{18}$$

Denoting the rows of the matrix $W$ by $W = (w_1, ..., w_d)^{\top}$, we can explicitly write the minimization expression as a function of $W$:

$$\hat{W} = \arg \min_{W} \sum_{t=1}^{d} H(w_t^{\top} X) \tag{19}$$

Then we can plug the MeanNN entropy estimator into Eq. (19) to obtain (after omitting irrelevant constants) an explicit contrast function to minimize:

$$\hat{W} = \arg \min_{W} S(W) = \arg \min_{W} \sum_{t=1}^{d} \sum_{i \neq j}^{n} \log((w_t^{\top}(x_i - x_j))^2) \tag{20}$$

The gradient of the contrast function $S(W)$ with respect to a rotation matrix $W$ may be found with the assistance of the so-called Givens rotations (see e.g. [14]). In this parametrization a rotation matrix $W \in R^{d \times d}$ is represented by a product of $d(d-1)/2$ plane rotations:

$$W = \prod_{s=1}^{d-1} \prod_{t=s+1}^{d} G_{st} \tag{21}$$

where $G_{st}$ is a rotation matrix corresponding to a rotation in the $st$ plane by an angle $\lambda_{st}$. It is the identity matrix except that its elements $(s,s),(s,t),(t,s),(t,t)$ form a two-dimensional (2-D) rotation matrix by

$$\begin{bmatrix} G_{st}(s,s) & G_{st}(s,t) \\ G_{st}(t,s) & G_{st}(t,t) \end{bmatrix} = \begin{bmatrix} \cos(\lambda_{st}) & \sin(\lambda_{st}) \\ -\sin(\lambda_{st}) & \cos(\lambda_{st}) \end{bmatrix} \tag{22}$$

The gradient of a single rotation matrix $G_{st}$ with respect to $\lambda_{st}$ is a zero matrix except for elements $(s,s),(s,t),(t,s),(t,t)$ for which

$$\frac{\partial}{\partial \lambda_{st}} \begin{bmatrix} G_{st}(s,s) & G_{st}(s,t) \\ G_{st}(t,s) & G_{st}(t,t) \end{bmatrix} = \begin{bmatrix} -\sin(\lambda_{st}) & \cos(\lambda_{st}) \\ -\cos(\lambda_{st}) & -\sin(\lambda_{st}) \end{bmatrix} \tag{23}$$

It can easily verified that the gradient of the contrast function (20) is given by

$$\frac{\partial}{\partial \lambda_{st}} S = \sum_{q,r=1}^{d} \frac{\partial S}{\partial w_{qr}} \frac{\partial w_{qr}}{\partial \lambda_{st}} = 2 \sum_{q,r=1}^{d} \sum_{i \neq j}^{n} \frac{(x_{ir} - x_{jr})}{|w_q^{\top}(x_i - x_j)|} \left[ \prod_{u=1}^{d-1} \prod_{v=u+1}^{d} \tilde{G}_{uv} \right]_{qr} \tag{24}$$

where $\tilde{G}_{uv} = \frac{\partial}{\partial \lambda_{uv}} G_{uv}$ if both $u = s$ and $v = t$, and $\tilde{G}_{uv} = G_{uv}$ otherwise.

The contrast function $S(W)$ and its gradient $\frac{\partial}{\partial \lambda_{st}} S$ may in theory suffer from discontinuities if a row $w_t$ is perpendicular to a vector $x_i - x_j$. To overcome this numerical difficulty we utilize a smoothed version of the contrast function $S(W, \epsilon)$ and give the expression for its gradient:

$$S(W, \epsilon) = \sum_{t=1}^{d} \sum_{i \neq j}^{n} \log((w_t^{\top}(x_i - x_j))^2 + \epsilon) \tag{25}$$

$$\frac{\partial}{\partial \lambda_{st}} S = \sum_{q,r=1}^{d} \frac{\partial S}{\partial w_{qr}} \frac{\partial w_{qr}}{\partial \lambda_{st}} = \sum_{q,r=1}^{d} \sum_{i \neq j}^{n} \frac{(x_{ir} - x_{jr})}{(w_q^{\top}(x_i - x_j))^2 + \epsilon} \left[ \prod_{u=1}^{d-1} \prod_{v=u+1}^{d} \tilde{G}_{uv} \right]_{qr} \tag{26}$$

For the optimization of the contrast function we apply the conjugate gradient method. The algorithm is summarized in Figure 1.

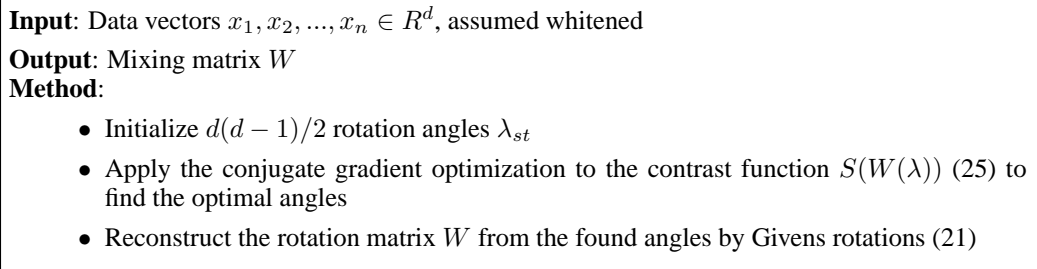

> **Input**: Data vectors $x_1, x_2, ..., x_n \in R^d$, assumed whitened
>
> **Output**: Mixing matrix $W$
> **Method**:
>
> - Initialize $d(d-1)/2$ rotation angles $\lambda_{st}$
> - Apply the conjugate gradient optimization to the contrast function $S(W(\lambda))$ (25) to find the optimal angles
> - Reconstruct the rotation matrix $W$ from the found angles by Givens rotations (21)

Figure 1: The MeanNN ICA algorithm

## 5 Experiments

First we study the set of 9 problems proposed by [2]. Each problem corresponds to a 1D probability distribution $q(x)$. One thousand pairs of random numbers $x$ and $y$ are mixed as $x' = x \cos \phi + y \sin \phi, y' = -x \sin \phi + y \cos \phi$ with random angle $\phi$ common to all pairs (i.e. $A$ is a pure rotation). We applied the conjugate gradient methods for the optimization of the contrast function (25) with $\epsilon = 1/n = 0.001$ in order to recover this rotation matrix. This was repeated 100 times with different angles $\phi$ and with different random sets of pairs $(x, y)$. To assess the quality of the estimator $\hat{A}$ (or, equivalently, of the back transformation $\hat{W} = \hat{A}^{-1}$), we use the Amari performance index $P_{err}$ from [1].

$$P_{err} = \frac{1}{2d} \sum_{i,j=1}^{d} \left( \frac{|p_{ij}|}{\max_k |p_{ik}|} + \frac{|p_{ij}|}{\max_k |p_{kj}|} \right) - 1 \qquad (27)$$

where $p_{ij} = (\hat{A}^{-1}A)_{ij}$. We compared our method with three state-of-the-art approaches: MILCA [16], RADICAL [13] and KernelICA [2]. We used the official code proposed by authors[1]. For the first two techniques that utilize different information theoretic measures assessed by order statistics it is highly recommended to use dataset augmentation. This is a computationally intensive technique for the dataset enlargement by replacing each data set point with a fixed number (usually 30) new data points randomly generated in the small neighborhood of the original point. The proposed method gives smooth results without any additional augmentation due to its smooth nature (see Eq. (13)).

| pdfs | MILCA | MILCA Aug | RADICAL | RADICAL Aug | KernelICA | MeanNN ICA |
|------|-------|-----------|---------|-------------|-----------|------------|
| a | 3.3 | 2.5 | 3.6 | 2.8 | 3.3 | **2.4** |
| b | 3.4 | 3.0 | 3.6 | 3.3 | 3.0 | **2.6** |
| c | 7.5 | 4.4 | 7.6 | 5.4 | 4.9 | **4.2** |
| d | 1.8 | 1.7 | **1.4** | 1.6 | 1.4 | 1.4 |
| e | 1.7 | 1.6 | 1.5 | 1.7 | 1.5 | **1.4** |
| f | 1.4 | **1.3** | 1.6 | 1.4 | 1.4 | 1.4 |
| g | 1.4 | **1.3** | 1.6 | 1.4 | 1.4 | 1.4 |
| h | 1.7 | 2.0 | 1.6 | 1.7 | **1.4** | 1.5 |
| i | 1.9 | 2.1 | 1.8 | 1.8 | **1.5** | 1.8 |

Table 2: Amari performance (multiplied by 100) for two-component ICA. The distributions are: (a) Student with 3 degrees of freedom; (b) double exponential; (c) Student with 5 degrees of freedom; (d) exponential; (e) mixture of two double exponentials; (f) symmetric mixtures of two Gaussians; (g) nonsymmetric mixtures of two Gaussians; (h) symmetric mixtures of four Gaussians; (i) non-symmetric mixtures of four Gaussians.

In the explored cases the proposed method achieves the level of a state-of-the-art performance. This is well explained by the inherent smoothness of MeanNN estimator, see Figure 2. Here we presented

the comparison of different contrast functions based on different order statistics estimators for a grid of possible rotations angles for the mixture of two exponentially distributed random variables (case $e$). The contrast function corresponding to the order statistics $k = 10$ generally coincides with the MILCA approach. Also the contrast function corresponding to the order statistics $k = 30 \simeq \sqrt{n}$ generally coincides with the RADICAL method. One may see that MeanNN ICA contrast function leads to much more robust prediction of the rotation angle. One should mention that the gradient based optimization enables to obtain the global optimum with high precision as opposed to MILCA and RADICAL schemes which utilize subspace grid optimization.

Application of the gradient based optimization schemes also leads to a computational advantage. The number of needed function evaluations was limited by 20 as opposed to 150 evaluations for grid optimization schemes MILCA and RADICAL.

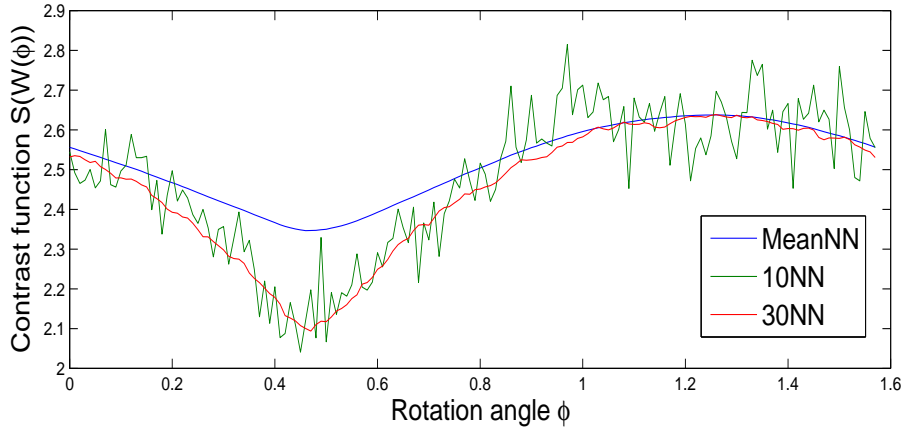

Figure 2: Convergence analysis for a mixture of two exponentially distributed random variables. Contrast function dependence on a rotation angle for different entropy estimators. 1000 samples, 0.01 radian grid.

We also studied the application of MeanNN ICA to multidimensional problems. For that purpose we chose at random $D$ (generally) different distributions, then we mixed them by a random rotation and ran the compared ICA algorithms to recover the rotation matrix. The results are presented at Table 3. MeanNN ICA achieved the best performance.

| dims | MILCA | MILCA Aug | RADICAL | RADICAL Aug | KernelICA | MeanNN ICA |
|------|-------|-----------|---------|-------------|-----------|------------|
| 2 | 3.0 | 3.3 | 3.1 | 3.0 | 2.9 | **2.5** |
| 4 | 2.7 | 2.7 | 2.8 | 2.3 | 2.6 | **2.2** |

Table 3: Amari index (multiplied by 100) for multidimensional ICA. 1000 samples, 10 repetitions

## 6 Conclusion

We proposed a novel approach for estimation of main information theoretic measures such as differential entropy, mutual information and divergence. The estimators represent smooth differential functions with clear geometrical meaning. Next this novel estimation technique was applied to the ICA problem. Compared to state-of-the-art ICA methods the proposed method demonstrated superior results in the conducted tests.

Studied state-of-the-art approaches can be divided in two groups. The first group is based on exact entropy estimation, that usually leads to high performance as demonstrated by MILCA and RADICAL. The drawback of such estimators is the lack of the gradient and therefore numerical difficulties in optimization. The second group apply different from entropy criteria, that benefit easy calculation of gradient (KernelICA). However such methods may suffer from deteriorated performance.

MeanNN ICA comprises the advantages of these two kinds of estimators. It represents a contrast function based on an accurate entropy estimation and its gradient is given analytically therefore it may be readily optimized.

Finally we mention that the proposed estimation method may further be applied to various problems in the field of machine learning and beyond.

## Footnotes

[1] http://www.klab.caltech.edu/~kraskov/MILCA/,      https://www.cs.umass.edu/~elm/ICA/, http://www.di.ens.fr/~fbach/kernel-ica/index.htm

## References

[1] S. Amari, A. Cichoki, and H.H.Yang. A new learning algorithm for blind signal separation. *Advances in Neural Information Processing Systems*, 8, 1996.

[2] F. Bach and M. Jordan. Kernel independent component analysis. *Journal of Machine Learning Research*, 3, 2002.

[3] A. J. Bell and T. J. Sejnowski. An information-maximization approach to blind separation and blind deconvolution. *Neural Computatiuon*, 7, 1995.

[4] J.-F. Cardoso. Multidimensional independent component analysis. *Proceedings of the International Conference on Acoustics, Speech, and Signal Processing (ICASSP'98)*, 1998.

[5] C.Jutten and J.Herault. Blind separation of sources, part 1: An adaptive algorithm based on neuromimetic architecture. *Signal Processing*, 1991.

[6] P. Comon. Independent component analysis, a new concept? *Signal Processing*, 36(3), 1994.

[7] Thomas M. Cover and Joy A. Thomas. *Elements of Information Theory*. Wiley-Interscience, August 1991.

[8] D.T.Pham and P.Garat. Blind separation of mixtures of independent signals through a quasi-maximum likelihood approach. *IEEE transactions on Signal Processing 45(7)*, 1997.

[9] A. Hyvarinen and E.Oja. A fast fixed point algorithm for independent component analysis. *Neural computation*, 9(7), 1997.

[10] A. Hyvarinen, J. Karhunen, and E. Oja. Independent component analysis. 2001.

[11] L. Kozachenko and N. Leonenko. On statistical estimation of entropy of random vector. *Problems Infor. Transmiss.*, 23 (2), 1987.

[12] A. Kraskov, H. Stögbauer, and P. Grassberger. Estimating mutual information. *Physical Review E*, 69:066138, 2004.

[13] E. Miller and J. Fisher. Ica using spacing estimates of entropy. *Proc. Fourth International Symposium on Independent Component Analysis and Blind Signal Separation, Nara, Japan, Apr. 2003, pp. 1047–1052.*, 2003.

[14] J. Peltonen and S. Kaski. Discriminative components of data. *IEEE Transactions on Neural Networks*, 16(1), 2005.

[15] H. Singh, N. Misra, V. Hnizdo, A. Fedorowicz, and Eugene Demchuk. Nearest neighbor estimates of entropy. *American Journal of Mathematical and Management Sciences*, 2003.

[16] H. Stögbauer, A. Kraskov, S. Astakhov, and P. Grassberger. Least-dependent-component analysis based on mutual information. *Phys. Rev. E*, 70(6):066123, Dec 2004.

[17] O. Vasicek. A test for normality based on sample entropy. *J. Royal Stat. Soc. B*, 38 (1):54–59, 1976.

[18] J. D. Victor. Binless strategies for estimation of information from neural data. *Physical Review*, 2002.

[19] Q. Wang, S. R. Kulkarni, and S. Verdu. A nearest-neighbor approach to estimating divergence between continuous random vectors. *IEEE Int. Symp. Information Theory, Seattle, WA*, 2006.

